# Counting function theorem for multi-layer networks

**Adam Kowalczyk**
Telecom Australia, Research Laboratories
770 Blackburn Road, Clayton, Vic. 3168, Australia
(a.kowalczyk@trl.oz.au)

## Abstract

We show that a randomly selected $N$-tuple $\vec{x}$ of points of $\mathbf{R}^n$ with probability $> 0$ is such that any multi-layer perceptron with the first hidden layer composed of $h_1$ threshold logic units can implement exactly $2 \sum_{i=0}^{h_1 n} \binom{N-1}{i}$ different dichotomies of $\vec{x}$. If $N > h_1 n$ then such a perceptron must have all units of the first hidden layer fully connected to inputs. This implies the maximal capacities (in the sense of Cover) of $2n$ input patterns per hidden unit and 2 input patterns per synaptic weight of such networks (both capacities are achieved by networks with single hidden layer and are the same as for a single neuron). Comparing these results with recent estimates of VC-dimension we find that in contrast to the single neuron case, for sufficiently large $n$ and $h_1$, the VC-dimension exceeds Cover's capacity.

## 1 Introduction

In the course of theoretical justification of many of the claims made about neural networks regarding their ability to learn a set of patterns and their ability to generalise, various concepts of maximal storage capacity were developed. In particular Cover's capacity [4] and VC-dimension [12] are two expressions of this notion and are of special interest here. We should stress that both capacities are not easy to compute and are presently known in a few particular cases of feedforward networks only. VC-dimension, in spite of being introduced much later, has been far

more researched, perhaps due to its significance expressed by a well known relation between generalisation and learning errors [12, 3]. Another reason why Cover's capacity gains less attention, perhaps, is that for the single neuron case it is twice higher than VC-dimension. Thus if one would hypothesise a similar relation to be true for other feedforward networks, he would judge Cover's capacity to be quite an unattractive parameter for generalisation estimates, where VC-dimension is believed to be unrealistically big. One of the aims of this paper is to show that this last hypothesis is not true, at least for some feedforward networks with sufficiently large number of hidden units. In the following we will always consider multilayer perceptrons with $n$ continuously-valued inputs, a single binary output, and one or more hidden layers, the first of which is made up of threshold logic units only.

The derivation of Cover's capacity for a single neuron in [4] is based on the so-called Function Counting Theorem, proved for the linear function in the sixties (c.f. [4]), which states that for an $N$-tuple $\vec{x}$ of points in general position one can implement $C(N, n) \stackrel{\text{def}}{=} 2 \sum_{i=0}^{n} \binom{N-1}{i}$ different dichotomies of $\vec{x}$. Extension of this result to the multilayer case is still an open problem (c.f. T. Cover's address at NIPS'92). One of the complications arising there is that in contrast to the single neuron case even for perceptrons with two hidden units the number of implementable dichotomies may be different for different $N$-tuples in general position [8]. Our first main result states that this dependence on $\vec{x}$ is relatively weak, that for a multilayer perceptron the number of implementable dichotomies (counting function) is constant on each of a finite number of connected components into which the space of $N$-tuples in general position can be decomposed. Then we show that for one of these components $C(N, nh_1)$ different dichotomies can be implemented, where $h_1$ is the number of hidden units in the first hidden layer (all assumed to be linear threshold logic units). This leads to an upper bound on Cover's capacity of $2n$ input patterns per (hidden) neuron and 2 patterns per adjustable synaptic weight, the same as for a single neuron. Comparing this result with a recent lower bound on VC-dimension of multilayer perceptrons [10] we find that for for sufficiently large $n$ and $h_1$ the VC-dimension is higher than Cover's capacity (by a factor $\log_2(h_1)$).

The paper extends some results announced in [5] and is an abbreviated version of a forthcoming paper [6].

## 2    Results

### 2.1    Standing assumptions and basic notation

We recall that in this paper *a multilayer perceptron* means a layered feedforward network with one or more hidden layers, and the first hidden layer built exclusively from threshold logic units.

A *dichotomy* of an $N$-tuple $\vec{x} = (x_1, ..., x_N) \in (\mathbf{R}^n)^N$ is a function $\delta : \{x_1, ..., x_N\} \to \{0, 1\}$. For a multilayer perceptron $F : \mathbf{R}^n \to \{0, 1\}$ let $\vec{x} \mapsto C_F(\vec{x})$ denote the number of different dichotomies of $\vec{x}$ which can be implemented for all possible selections of synaptic weights and biases. We shall call $C_F(\vec{x})$ a *counting function* following the terminology used in [4].

**Example 1.** $C_\phi(\vec{x}) = C(N, n) \stackrel{\text{def}}{=} 2 \sum_{i=0}^{n} \binom{N-1}{i}$ for a single threshold logic unit $\phi : \mathbf{R}^n \to \{0, 1\}$ [4]. $\square$

Points of an $N$-tuple $\vec{x} \in (\mathbf{R}^n)^N$ are said to be in *general position* if there does not exist an $l \stackrel{\text{def}}{=} \min(N, n-1)$-dimensional affine hyperplane in $\mathbf{R}^n$ containing $(l+2)$ of them. We use a symbol $\mathcal{GP}(n, N) \subset (\mathbf{R}^n)^N$ to denote that set of all $N$-tuples $\vec{x}$ in general position.

Throughout this paper we assume to be given a probability measure $d\mu \stackrel{\text{def}}{=} f \, dx$ on $\mathbf{R}^n$ such that the density $f : \mathbf{R}^n \to \mathbf{R}$ is a continuous function.

## 2.2   Counting function is locally constant

We start with a basic characterisations of the subset $\mathcal{GP}(n, N) \subset (\mathbf{R}^n)^N$.

**Theorem 1** (*i*) $\mathcal{GP}(n, N)$ *is an open and dense subset of* $(\mathbf{R}^n)^N$ *with a finite number of connected components.*

(*ii*) *Any of these components is unbounded, has an infinite Lebesgue measure and has a positive probability measure.*

**Proof outline.** (*i*) The key point to observe is that $\mathcal{GP}(n, N) = \{\vec{x} \; : \; p(\vec{x}) \neq 0\}$, where $p : (\mathbf{R}^n)^N \to \mathbf{R}$ is a polynomial on $(\mathbf{R}^n)^N$. This implies immediately that $\mathcal{GP}(n, N)$ is open and dense in $(\mathbf{R}^n)^N$. The finite number of connected components follows from the results of Milnor [7] (c.f. [2]).

(*ii*) This follows from an observation that each of the connected components $C_i$ has the property that if $(x_1, ..., x_N) \in C_i$ and $a > 0$, then $(ax_1, ..., ax_N) \in C_i$. $\square$

As Example 1 shows, for a single neuron the counting function is constant on $\mathcal{GP}(n, N)$. However, this may not be the case even for perceptrons with two hidden units and two inputs (c.f. [8, 6] for such examples and Corollary 8). Our first main result states that this dependence on $\vec{x}$ is relatively weak.

**Theorem 2** $C_F(\mathbf{x})$ *is constant on connected components of* $\mathcal{GP}(n, N)$.

**Proof outline.** The basic heuristic behind the proof of this theorem is quite simple. If we have an $N$-tuple $\vec{x} \in (\mathbf{R}^n)^N$ which is split into two parts by a hyperplane, then this split is preserved for any sufficiently small perturbation $\vec{y} \in (\mathbf{R}^n)^N$ of $\vec{x}$, and vice versa, any split of $\vec{y}$ corresponds to a split of $\vec{x}$. The crux is to show that if $\vec{x}$ is in general position, then a minute perturbation $\vec{y}$ of $\vec{x}$ cannot allow a bigger number of splits than is possible for $\vec{x}$. We refer to [6] for details. $\square$

The following corollary outlines the main impact of Theorem 2 on the rest of the paper. It reduces the problem of investigation of the function $C_F(\mathbf{x})$ on $\mathcal{GP}(n, N)$ to a consideration of a set of individual, special cases of $N$-tuples which, in particular, are amenable to be solved analytically.

**Corollary 3** *If* $\vec{x} \in \mathcal{GP}(n, N)$, *then* $C_F(\vec{x}) = C_F(\vec{y})$ *for a randomly selected $N$-tuple $\vec{y} \in (\mathbf{R}^n)^N$ with a probability $> 0$.*

## 2.3  A case of special component of $\mathcal{GP}(n, N)$

The following theorem is the crux of the paper.

**Theorem 4** *There exists a connected component $CC \subset \mathcal{GP}(n, N) \subset (\mathbf{R}^n)^N$ such that*

$$C_F(\vec{x}) = C(N, nh_1) = 2 \sum_{i=0}^{h_1 n} \binom{N-1}{i} \qquad (\text{for } \vec{x} \in CC)$$

*with equality iff the input and first hidden layer are fully connected. The synaptic weights to units not in the first hidden layer can be constant.*

Using now Corollary 3 we obtain:

**Corollary 5** $C_F(\vec{x}) = C(N, nh_1)$ *for $\vec{x} \in (\mathbf{R}^n)^N$ with a probability $> 0$.*

The component $CC \subset \mathcal{GP}(n, N)$ in Theorem 4 is defined as the connected component containing

$$\vec{p}_N \stackrel{\text{def}}{=} (\mathbf{c}(t_1), \mathbf{c}(t_2), ..., \mathbf{c}(t_N)) \in (\mathbf{R}^n)^N, \tag{1}$$

where $\mathbf{c} : \mathbf{R} \to \mathbf{R}^n$ is the curve defined as $\mathbf{c}(t) \stackrel{\text{def}}{=} (t, t^2, ..., t^n)$ for $t \in \mathbf{R}$ and $0 < t_1 < t_2 < \cdots < t_N$ are some numbers (this example has been considered previously in [11]). The essential part of the proof of Theorem 4 is showing the basic properties of the $N$-tuple $\vec{p}_N$ which will be described by the Lemma below.

Any dichotomy $\delta$ of the $N$-tuple $\vec{p}_N$ (c.f. 1) is uniquely defined by its value at $\mathbf{c}(t_1)$ (2 options) and the set of indices $1 \le i_1 < i_2 < \cdots < i_k < N$ of all *transitional pairs* $(\mathbf{c}(t_{i_j}), \mathbf{c}(t_{i_j+1}))$, i.e. all indices $i_j$ such that $\delta(\mathbf{c}(t_{i_j})) \ne \delta(\mathbf{c}(t_{i_j+1}))$, where $j = 1, ..., k$, (additional $\binom{N-1}{k}$ options). Thus it is easily seen that there exist altogether $2\binom{N-1}{k}$ different dichotomies of $\vec{p}_N$ for any given number $k$ of transitional pairs, where $0 \le k < N$.

**Lemma 6** *Given integers $n, N, h > 0$, $k \ge 0$ and a dichotomy $\delta$ of $\vec{p}_N$ with $k$ transitional pairs.*

*(i) If $k \le nh$, then there exist hyperplanes $H_{(\mathbf{w}_i, b_i)}$, $(\mathbf{w}_i, b_i) \in \mathbf{R}^n \times \mathbf{R}$, such that*

$$\delta(\mathbf{p}_j) = \theta \left( b_0 + \sum_{i=1}^{h} v_i \theta(\mathbf{w}_i \cdot \mathbf{p}_j + b_i) \right), \tag{2}$$

$$\mathbf{w}_i \cdot \mathbf{p}_j + b_i \ne 0, \tag{3}$$

*for $i = 1, ..., h$ and $j = 1, ..., N$; here $v_i \stackrel{\text{def}}{=} 1$ if $n$ is even and $v_i \stackrel{\text{def}}{=} (-1)^i$ if $n$ is odd, $b_0 \stackrel{\text{def}}{=} -0.5$ if $n$ is odd, $h$ is even and $\delta(\mathbf{p}_0) = 1$, and $b_0 \stackrel{\text{def}}{=} 0.5$, otherwise.*

*(ii) If $k = nh$, then $w_{ij} \ne 0$ for $j = 1, ..., n$ and $i = 1, ..., h$, where $\mathbf{w}_i = (w_{i1}, w_{i2}, ..., w_{in})$.*

*(iii) If $k > nh$, then (2) and (3) cannot be satisfied.*

The proof of Lemma 6 relies on usage of the Vandermonde determinant and its derivatives. It is quite technical and thus not included here (c.f. [6] for details).

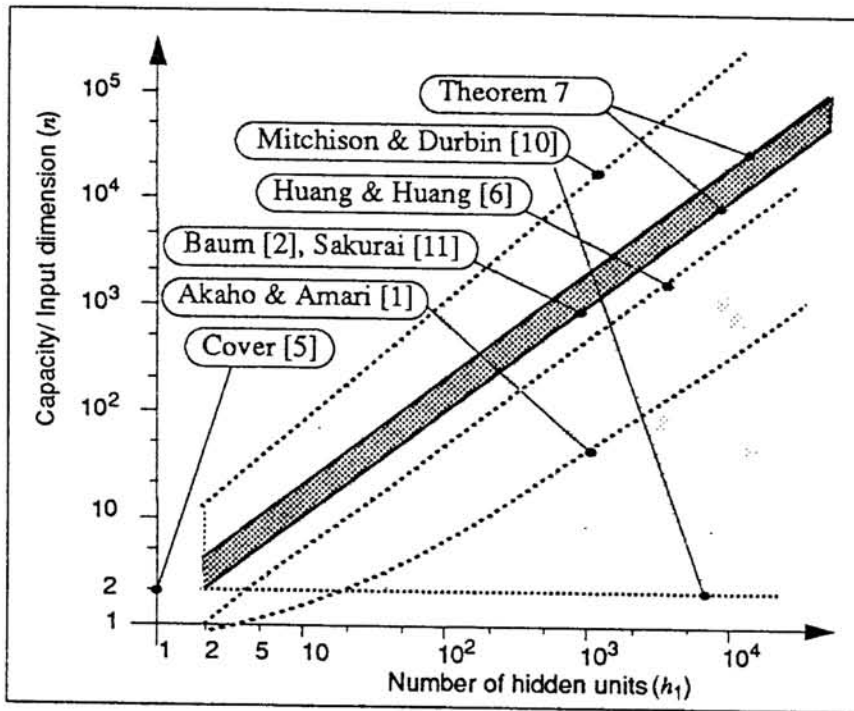

Figure 1: Some estimates of capacity.

## 3 Discussion

### 3.1 An upper bound on Cover's capacity

The *Cover's capacity* (or just *capacity*) of a neural network $F : \mathbf{R}^n \to \{0, 1\}$, $Cap(F)$, is defined as the maximal $N$ such that for a randomly selected $N$-tuple $\vec{x} = (x_1, ..., x_N) \in (\mathbf{R}^n)^N$ of points of $\mathbf{R}^n$, the network can implement $1/2$ of all dichotomies of $\vec{x}$ with probability 1 [4, 8].

Corollary 5 implies that $Cap(F)$ is not greater than maximal $N$ such that

$$C_F(\vec{p}_N)/2^N = C(N, nh_1) \geq 1/2. \tag{4}$$

since any property which holds with probability 1 on $(\mathbf{R}^n)^N$ must also hold probability 1 on $CC$ (c.f Theorem 4). The left-hand-side of the above equation is just the sum of the binomial expansion of $(1/2 + 1/2)^{N-1}$ up to $h_1 n$-th term, so, using the symmetry argument, we find that it is $\geq 1/2$ if and only if it has at least half of the all terms, i.e. when $N - 1 + 1 \leq 2(h_1 n + 1)$. Thus the $2(h_1 n + 1)$ is the maximal value of $N$ satisfying (4). [1] Now let us recall that a multilayer perceptron as in this paper can implement any dichotomy of any $N$-tuple $\vec{x}$ in general position if $N \leq nh_1 + 1$ [1, 11]. This leads to the following result:

**Theorem 7**

$$nh_1 + 1 \leq Cap(F) \leq 2(nh_1 + 1).$$

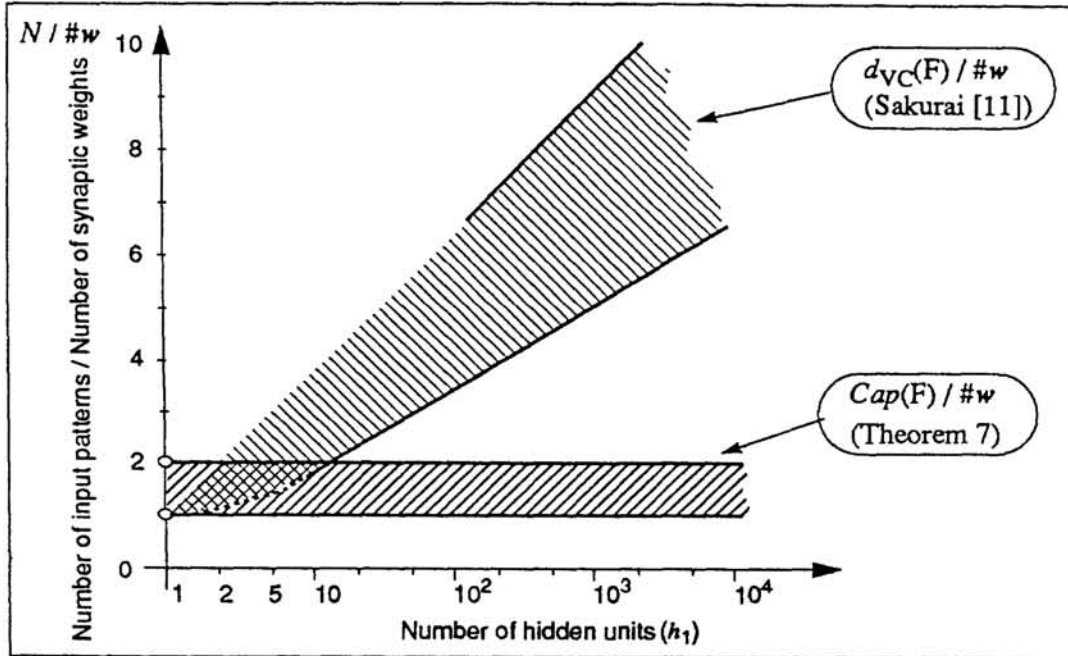

Figure 2: Comparison of estimates of the ratios of Cover's capacity per synaptic weight $(Cap(F)/\#\mathbf{w})$ and VC-dimension per synaptic weight $(d_{VC}(F)/\#\mathbf{w})$. (Note that the upper bound for VC-dimension has so far been proved for low number of hidden layers [9,10].)

for any multilayer perceptron $F : \mathbf{R}^n \rightarrow \{0, 1\}$ with the first hidden layer built from the $h_1$ threshold logic units. For the most efficient networks in this class, with a single hidden layer, we thus obtain the following result:

$$1 - O(1/nh_1) \leq Cap(F)/\#\mathbf{w} \leq 2,$$

where $\#\mathbf{w}$ denotes the number of synaptic weights and biases.

### 3.2    A relation to VC-dimension

The *VC-dimension*, $d_{VC}(F)$, is defined as the largest $N$ such that there exists an $N$-tuple $\vec{\mathbf{x}} = (\mathbf{x}_1, ..., \mathbf{x}_N) \in (\mathbf{R}^n)^N$ for which the network can implement all possible $2^N$ dichotomies. Recent results of Sakurai [10] imply

$$d_{VC}(F) \geq (1/2)nh_1(\log_2 h_1 + o(\log_2 h_1) + O((\log_2 h_1)^2/n)). \tag{5}$$

For sufficiently large $n$ and $h_1$ this estimate exceeds $2(nh_1 + 1)$ which is an upper bound on $Cap(F)$. Thus, in contrast to the single threshold logic unit case we have the following (c.f. Fig. 3):

**Corollary 8** $Cap(F) < d_{VC}(F)$ *if* $h_1 \gg 1$.

### 3.3    Memorisation ability of multilayer perceptron

Corollary 8 combined with Theorem 7 and Figure 2 imply that for some cases of patters in general position multilayer perceptron can memorise and reliably retrieve

(even with 100% accuracy) much more ($\approx \log_2(h_1)$ times more) than 2 patterns per connection, as is the case for a single neuron [4]. This proves that co-operation between hidden units can significantly improve the storage efficiency of neural networks.

## 3.4  A relation to PAC learning

Vapnik's estimate of *generalisation error* [12] (an error rate on independent test set)

$$E_G(F) \le E_L(F) + D(N, d_{VC}(F), E_L, \eta) \tag{6}$$

holds for $N > d_{VC}(F)$ with probability larger that $(1 - \eta)$. It contains two terms: (*i*) *learning error* $E_L(F)$ and (*ii*) *confidence interval*

$$D(p, d_{VC}, E_L, \eta) \stackrel{\text{def}}{=} 2\Psi(p, d_{VC}, \eta) \left[ 1 + \sqrt{1 + E_L/\Psi(p, d_{VC}, \eta)} \right],$$

where

$$\Psi(N, d_{VC}, \eta) = (\ln \frac{2N}{d_{VC}} + 1)\frac{d_{VC}}{2N} - \frac{\ln \eta}{N}.$$

The ability of obtaining small learning error $E_L(F)$ is, in a sense, controlled by $Cap(F)$, while the size of the confidence interval $D$ is controlled by both $d_{VC}(F)$ and $Cap(F)$ (through $E_L(F)$). For a multilayer perceptron as in Theorem 7 when $d_{VC}(F) >> Cap(F)$ (Fig. 2) it can turn out that actually the capacity rather than the VC-dimension is the most critical factor in obtaining low generalisation error $E_G(F)$. This obviously warrants further research into the relation between capacity and generalisation.

The theoretical estimates of generalisation error based on VC-dimension are believed to be too pessimistic in comparison with some experiments. One may hypothesise that this is caused by too high values of $d_{VC}(F)$ used in estimates such as (6). Since Cover's capacity in the case multilayer perceptron with $h_1 >> 1$ turned up to be much lower than VC-dimension, one may hope that more realistic estimates could be achieved with generalisation estimates linked directly to capacity. This subject will obviously require further research. Note that some results along these lines can be found in Cover's paper [4].

## 3.5  Some open problems

Theorem 7 gives estimates of capacity per variable connection for a network with the minimal number of neurons in the first hidden layer showing that these neurons have to be fully connected. The natural question arises at this point as to whether a network with a bigger number but not fully connected neurons in the first hidden layer can achieve a better capacity (per adjustable synaptic weight).

The values of the counting function $\vec{x} \mapsto C_F(\vec{x})$ are provided in this paper for the particular class of points in general position, for $\vec{x} \in CC \subset (\mathbf{R}^n)^N$. The natural question is whether they may be by chance a lower or upper bound for the counting function for the general case of $\vec{x} \in (\mathbf{R}^n)^N$ ? The results of Sakurai [11] seem to point to the former case: in his case, the sequences $\vec{p}_N = (p_1, ..., p_N)$ turned out to be "the hardest" in terms of hidden units required to implement 100% of

dichotomies. Corollary 8 and Figure 1 also support this lower bound hypothesis. They imply in particular that there exists a $N'$-tuple $\vec{y} = (y_1, y_2, ..., y_{N'}) \in (\mathbf{R}^n)^{N'}$, where $N' \stackrel{\text{def}}{=}$ VC-dimension $> N$, such that $C_F(\vec{y}) = 2^{N'} \gg 2^N > C_F(\vec{p}_N)$ for sufficiently large $n$ and $h$.

## 4 Acknowledgement

The permission of Managing Director, Research and Information Technology, Telecom Australia, to publish this paper is gratefully acknowledged.

## Footnotes

[1]Note that for large $N$ the choice of cutoff value $1/2$ is not critical, since the probability of a dichotomy being implementable drops rapidly as $h_1 n$ approaches $2^N/2$.

## References

[1] E. Baum. On the capabilities of multilayer perceptrons. *Journal of Complexity*, 4:193–215, 1988.

[2] S. Ben-David and M. Lindenbaum. Localization vs. identification of semi-algebraic sets. In *Proceedings of the Sixth Annual Workshop on Computational Learning Theory (to appear)*, 1993.

[3] A. Blumer, A. Ehrenfeucht, D. Haussler, and M.K. Warmuth. Learnability and the Vapnik-Chernovenkis dimensions. *Journal of the ACM*, 36:929–965, (Oct. 1989).

[4] T.M. Cover. Geometrical and statistical properties of linear inequalities with applications to pattern recognition. *IEEE Trans. Elec. Comp.*, EC-14:326–334, 1965.

[5] A. Kowalczyk. Some estimates of necessary number of connections and hidden units for feed-forward networks. In S.J. Hanson *et al.*, editor, *Advances in Neural Information Processing Systems*, volume 5. Morgan Kaufman Publishers, Inc., 1992.

[6] A. Kowalczyk. Estimates of storage capacity of multi-layer perceptron with threshold logic hidden units. In preparation, 1994.

[7] J. Milnor. On Betti numbers of real varieties. *Proceedings of AMS*, 15:275–280, 1964.

[8] G.J. Mitchison and R.M. Durbin. Bounds on the learning capacity of some multi-layer networks. *Biological Cybernetics*, 60:345–356, (1989).

[9] A. Sakurai. On the VC-dimension of depth four threshold circuits and the complexity of boolean-valued functions. Manuscript, Advanced Research Laboratory, Hitachi Ltd., 1993.

[10] A. Sakurai. Tighter bounds of the VC-dimension of three-layer networks. In *WCNN93*, 1993.

[11] A. Sakurai. n-h-1 networks store no less $n \cdot h + 1$ examples but sometimes no more. In *Proceedings of IJCNN92*, pages III–936–III–941. IEEE, June 1992.

[12] V. Vapnik. *Estimation of Dependences Based on Empirical Data*. Springer-Verlag, 1982.